# Optimal Manifold Representation of Data: An Information Theoretic Approach

**Denis Chigirev and William Bialek**
Department of Physics and the Lewis-Sigler Institute for Integrative Genomics
Princeton University, Princeton, New Jersey 08544
`chigirev,wbialek@princeton.edu`

## Abstract

We introduce an information theoretic method for nonparametric, non-linear dimensionality reduction, based on the infinite cluster limit of rate distortion theory. By constraining the information available to manifold coordinates, a natural probabilistic map emerges that assigns original data to corresponding points on a lower dimensional manifold. With only the information-distortion trade off as a parameter, our method determines the shape of the manifold, its dimensionality, the probabilistic map and the prior that provide optimal description of the data.

## 1 A simple example

Some data sets may not be as complicated as they appear. Consider the set of points on a plane in Figure 1. As a two dimensional set, it requires a two dimensional density $\rho(x, y)$ for its description. Since the data are sparse the density will be almost singular. We may use a smoothing kernel, but then the data set will be described by a complicated combination of troughs and peaks with no obvious pattern and hence no ability to generalize. We intuitively, however, see a strong one dimensional structure (a curve) underlying the data. In this paper we attempt to capture this intuition formally, through the use of the infinite cluster limit of rate distortion theory.

Any set of points can be embedded in a hypersurface of any intrinsic dimensionality if we allow that hypersurface to be highly "folded." For example, in Figure 1, any curve that goes through all the points gives a one dimensional representation. We would like to avoid such solutions, since they do not help us discover structure in the data. Looking for a simpler description one may choose to penalize the curvature term [1]. The problem with this approach is that it is not easily generalized to multiple dimensions, and requires the dimensionality of the solution as an input.

An alternative approach is to allow curves of all shapes and sizes, but to send the reduced coordinates through an information bottleneck. With a fixed number of bits, position along a highly convoluted curve becomes uncertain. This will penalize curves that follow the data too closely (see Figure 1). There are several advantages to this approach. First, it removes the artificiality introduced by Hastie [2] of adding to the cost function only orthogonal errors. If we believe that data points fall out of the manifold due to noise, there is no reason to treat the projection onto the manifold as exact. Second, it does not require the dimension-

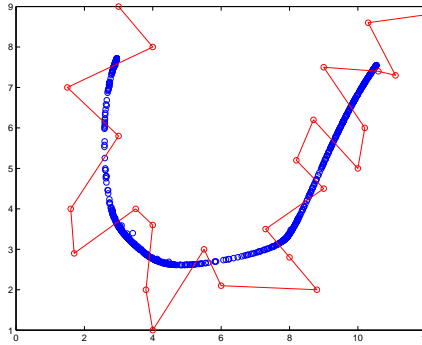

Figure 1: Rate distortion curve for a data set of 25 points (red). We used 1000 points to represent the curve which where initialized by scattering them uniformly on the plane. Note that the produced curve is well defined, one dimensional and smooth.

ality of the solution manifold as an input. By adding extra dimensions, one quickly looses the precision with which manifold points are specified (due to the fixed information bottleneck). Hence, the optimal dimension emerges naturally. This also means that the method works well in many dimensions with no adjustments. Third, the method handles sparse data well. This is important since in high dimensional spaces all data sets are sparse, i.e. they look like points in Figure 1, and the density estimation becomes impossible. Luckily, if the data are truly generated by a lower dimensional process, then density estimation in the data space is not important (from the viewpoint of prediction or any other). What is critical is the density of the data along the manifold (known in latent variable modeling as a prior), and our algorithm finds it naturally.

## 2  Latent variable models and dimensionality reduction

Recently, the problem of reducing the dimensionality of a data set has received renewed attention [3,4]. The underlying idea, due to Hotelling [5], is that most of the variation in many high dimensional data sets can often be explained by a few latent variables. Alternatively, we say that rather than filling the whole space, the data lie on a lower dimensional manifold. The dimensionality of this manifold is the dimensionality of the latent space and the coordinate system on this manifold provides the latent variables.

Traditional tools of principal component analysis (PCA) and factor analysis (FA) are still the most widely used methods in data analysis. They project the data onto a hyperplane, so the reduced coordinates are easy to interpret. However, these methods are unable to deal with nonlinear correlations in a data set. To accommodate nonlinearity in a data set, one has to relax the assumption that the data is modeled by a hyperplane, and allow a general low dimensional manifold of unknown shape and dimensionality. The same questions that we asked in the previous section apply here. What do we mean by requiring that "the manifold models the data well"? In the next section, we formalize this notion by defining the manifold description of data as a doublet (the shape of the manifold and the projection map). Note that we do not require the probability distribution over the manifold (known for generative models [6,7] as a prior distribution over the latent variables and postulated a priori). It is completely determined by the doublet.

Nonlinear correlations in data can also be accommodated implicitly, without constructing an actual low dimensional manifold. By mapping the data from the original space to an even higher dimensional feature space, we may hope that the correlations will become linearized and PCA will apply. Kernel methods [8] allow us to do this without actually constructing an explicit map to feature space. They introduce nonlinearity through an a priori nonlinear kernel. Alternatively, autoassociative neural networks [9] force the data through a bottleneck (with an internal layer of desired dimensionality) to produce a reduced

description. One of the disadvantages of these methods is that the results are not easy to interpret.

Recent attempts to describe a data set with a low dimensional representation generally follow into two categories: spectral methods and density modeling methods. Spectral methods (LLE [3], ISOMAP [4], Laplacian eigenmaps [10]) give reduced coordinates of an a priori dimensionality by introducing a quadratic cost function in reduced coordinates (hence eigenvectors are solutions) that mimics the relationships between points in the original data space (geodesic distance for ISOMAP, linear reconstruction for LLE). Density modeling methods (GTM [6], GMM [7]) are generative models that try to reproduce the data with fewer variables. They require a prior and a parametric generative model to be introduced a priori and then find optimal parameters via maximum likelihood.

The approach that we will take is inspired by the work of Kramer [9] and others who tried to formulate dimensionality reduction as a compression problem. They tried to solve the problem by building an explicit neural network encoder-decoder system which restricted the information implicitly by limiting the number of nodes in the bottleneck layer. Extending their intuition with the tools of information theory, we recast dimensionality reduction as a compression problem where the bottleneck is the information available to manifold coordinates. This allows us to define the optimal manifold description as that which produces the best reconstruction of the original data set, given that the coordinates can only be transmitted through a channel of fixed capacity.

## 3 Dimensionality reduction as compression

Suppose that we have a data set $X$ in a high dimensional state space $R^D$ described by a density function $\rho(\boldsymbol{x})$. We would like to find a "simplified" description of this data set. One may do so by visualizing a lower dimensional manifold $\mathcal{M}$ that "almost" describes the data. If we have a manifold $\mathcal{M}$ and a stochastic map $P_{\mathcal{M}} : \boldsymbol{x} \to P_{\mathcal{M}}(\boldsymbol{\mu}|\boldsymbol{x})$ to points $\boldsymbol{\mu}$ on the manifold, we will say that they provide a *manifold description* of the data set $X$. Note that the stochastic map here is well justified: if a data point does not lie exactly on the manifold then we should expect some uncertainty in the estimation of the value of its latent variables. Also note that we do not need to specify the inverse (generative) map: $\mathcal{M} \to R^D$; it can be obtained by Bayes' rule.

The manifold description $(\mathcal{M}, P_{\mathcal{M}})$ is a less than faithful representation of the data. To formalize this notion we will introduce the *distortion* measure $D(\mathcal{M}, P_{\mathcal{M}}, \rho)$:

$$D(\mathcal{M}, P_{\mathcal{M}}, \rho) = \int_{x \in R^D} \int_{\mu \in \mathcal{M}} \rho(\boldsymbol{x}) P_{\mathcal{M}}(\boldsymbol{\mu}|\boldsymbol{x}) \|\boldsymbol{x} - \boldsymbol{\mu}\|^2 d^D x D\mu. \tag{1}$$

Here we have assumed the Euclidean distance function for simplicity.

The stochastic map, $P_{\mathcal{M}}(\boldsymbol{\mu}|\boldsymbol{x})$, together with the density, $\rho(\boldsymbol{x})$, define a joint probability function $P(\mathcal{M}, X)$ that allows us to calculate the mutual information between the data and its manifold representation:

$$I(X, \mathcal{M}) = \int_{x \in X} \int_{\mu \in \mathcal{M}} P(\boldsymbol{x}, \boldsymbol{\mu}) \log \left[ \frac{P(\boldsymbol{x}, \boldsymbol{\mu})}{\rho(\boldsymbol{x}) P_{\mathcal{M}}(\boldsymbol{\mu})} \right] d^D x D\mu. \tag{2}$$

This quantity tells us how many bits (on average) are required to encode $\boldsymbol{x}$ into $\boldsymbol{\mu}$. If we view the manifold representation of $X$ as a compression scheme, then $I(X, \mathcal{M})$ tells us the necessary capacity of the channel needed to transmit the compressed data.

Ideally, we would like to obtain a manifold description $\{\mathcal{M}, P_{\mathcal{M}}(\mathcal{M}|X)\}$ of the data set $X$ that provides both a low distortion $D(\mathcal{M}, P_{\mathcal{M}}, \rho)$ and a good compression (i.e. small

$I(X, \mathcal{M})$). The more bits we are willing to provide for the description of the data, the more detailed a manifold that can be constructed. So there is a trade off between how faithful a manifold representation can be and how much information is required for its description. To formalize this notion we introduce the concept of an *optimal manifold*.

DEFINITION. Given a data set $X$ and a channel capacity $I$, a manifold description $(\mathcal{M}, P_{\mathcal{M}}(\mathcal{M}|X))$ that minimizes the distortion $D(\mathcal{M}, P_{\mathcal{M}}, X)$, and requires only information $I$ for representing an element of $X$, will be called an *optimal manifold $\mathcal{M}(I, X)$*.

Note that another way to define an optimal manifold is to require that the information $I(\mathcal{M}, X)$ is minimized while the average distortion is fixed at value $D$. The shape and the dimensionality of optimal manifold depends on our information resolution (or the description length that we are willing to allow). This dependence captures our intuition that for real world, multi-scale data, a proper manifold representation must reflect the compression level we are trying to achieve.

To find the optimal manifold $(\mathcal{M}(I), P_{\mathcal{M}(I)})$ for a given data set $X$, we must solve a constrained optimization problem. Let us introduce a Lagrange multiplier $\lambda$ that represents the trade off between information and distortion. Then optimal manifold $\mathcal{M}(I)$ minimizes the functional:

$$\mathcal{F}(\mathcal{M}, P_{\mathcal{M}}) = D + \lambda I. \tag{3}$$

Let us parametrize the manifold $\mathcal{M}$ by $\boldsymbol{t}$ (presumably $\boldsymbol{t} \in R^d$ for some $d \leq D$). The function $\boldsymbol{\gamma}(\boldsymbol{t}) : \boldsymbol{t} \to \mathcal{M}$ maps the points from the parameter space onto the manifold and therefore describes the manifold. Our equations become:

$$D = \int\int d^D x \, d^d t \, \rho(\boldsymbol{x}) P(\boldsymbol{t}|\boldsymbol{x}) \|\boldsymbol{x} - \boldsymbol{\gamma}(\boldsymbol{t})\|^2, \tag{4}$$

$$I = \int\int d^D x \, d^d t \, \rho(\boldsymbol{x}) P(\boldsymbol{t}|\boldsymbol{x}) \log \frac{P(\boldsymbol{t}|\boldsymbol{x})}{P(\boldsymbol{t})}, \tag{5}$$

$$\mathcal{F}(\boldsymbol{\gamma}(\boldsymbol{t}), P(\boldsymbol{t}|\boldsymbol{x})) = D + \lambda I. \tag{6}$$

Note that both information and distortion measures are properties of the manifold description doublet $\{\mathcal{M}, P_{\mathcal{M}}(\mathcal{M}|X)\}$ and are invariant under reparametrization. We require the variations of the functional to vanish for optimal manifolds $\delta\mathcal{F}/\delta\boldsymbol{\gamma}(\boldsymbol{t}) = 0$ and $\delta\mathcal{F}/\delta P(\boldsymbol{t}|\boldsymbol{x}) = 0$, to obtain the following set of self consistent equations:

$$P(\boldsymbol{t}) = \int d^D x \, \rho(\boldsymbol{x}) P(\boldsymbol{t}|\boldsymbol{x}), \tag{7}$$

$$\boldsymbol{\gamma}(\boldsymbol{t}) = \frac{1}{P(\boldsymbol{t})} \int d^D x \, \boldsymbol{x} \rho(\boldsymbol{x}) P(\boldsymbol{t}|\boldsymbol{x}), \tag{8}$$

$$P(\boldsymbol{t}|\boldsymbol{x}) = \frac{P(\boldsymbol{t})}{\Pi(\boldsymbol{x})} e^{-\frac{1}{\lambda}\|\boldsymbol{x}-\boldsymbol{\gamma}(\boldsymbol{t})\|^2}, \tag{9}$$

$$\Pi(\boldsymbol{x}) = \int d^d t \, P(\boldsymbol{t}) e^{-\frac{1}{\lambda}\|\boldsymbol{x}-\boldsymbol{\gamma}(\boldsymbol{t})\|^2}. \tag{10}$$

In practice we do not have the full density $\rho(\boldsymbol{x})$, but only a discrete number of samples. So we have to approximate $\rho(\boldsymbol{x}) = \frac{1}{N} \sum \delta(\boldsymbol{x} - \boldsymbol{x}_i)$, where $N$ is the number of samples, $i$ is the sample label, and $\boldsymbol{x}_i$ is the multidimensional vector describing the $i$th sample.

Similarly, instead of using a continuous variable $t$ we use a discrete set $t \in \{t_1, t_2, ..., t_K\}$ of $K$ points to model the manifold. Note that in $(7 - 10)$ the variable $t$ appears only as an argument for other functions, so we can replace the integral over $t$ by a sum over $k = 1..K$. Then $P(t|x)$ becomes $P_k(x_i)$, $\gamma(t)$ is now $\gamma_k$, and $P(t)$ is $P_k$. The solution to the resulting set of equations in discrete variables $(11 - 14)$ can be found by an iterative Blahut-Arimoto procedure [11] with an additional EM-like step. Here $(n)$ denotes the iteration step, and $\alpha$ is a coordinate index in $R^D$. The iteration scheme becomes:

$$P_k^{(n)} = \frac{1}{N} \sum_{i=1}^{N} P_k^{(n)}(x_i) \tag{11}$$

$$\gamma_{k,\alpha}^{(n)} = \frac{1}{P_k^{(n)}} \frac{1}{N} \sum_{i=1}^{N} x_{i,\alpha} P_k^{(n)}(x_i), \tag{12}$$

$$\text{where} \quad \alpha = 1, \ldots, D,$$

$$\Pi^{(n)}(x_i) = \sum_{k=1}^{K} P_k^{(n)} e^{-\frac{1}{\lambda} \|x_i - \gamma_k^{(n)}\|^2} \tag{13}$$

$$P_k^{(n+1)}(x_i) = \frac{P_k^{(n)}}{\Pi^{(n)}(x_i)} e^{-\frac{1}{\lambda} \|x_i - \gamma_k^{(n)}\|^2}. \tag{14}$$

One can initialize $\gamma_k^0$ and $P_k^0(x_i)$ by choosing $K$ points at random from the data set and letting $\gamma_k = x_{i(k)}$ and $P_k^0 = 1/K$, then use equations (13) and (14) to initialize the association map $P_k^0(x_i)$. The iteration procedure $(11 - 14)$ is terminated once

$$\max_k |\gamma_k^n - \gamma_k^{n-1}| < \epsilon, \tag{15}$$

where $\epsilon$ determines the precision with which the manifold points are located. The above algorithm requires the information distortion cost $\lambda = -\delta D / \delta I$ as a parameter. If we want to find the manifold description $(\mathcal{M}, P(\mathcal{M}|X))$ for a particular value of information $I$, we can plot the curve $I(\lambda)$ and, because it's monotonic, we can easily find the solution iteratively, arbitrarily close to a given value of $I$.

## 4  Evaluating the solution

The result of our algorithm is a collection of $K$ manifold points, $\gamma_k \in \mathcal{M} \subset R^D$, and a stochastic projection map, $P_k(x_i)$, which maps the points from the data space onto the manifold. Presumably, the manifold $\mathcal{M}$ has a well defined intrinsic dimensionality $d$. If we imagine a little ball of radius $r$ centered at some point on the manifold of intrinsic dimensionality $d$, and then we begin to grow the ball, the number of points on the manifold that fall inside will scale as $r^d$. On the other hand, this will not be necessarily true for the original data set, since it is more spread out and resembles locally the whole embedding space $R^D$. The Grassberger-Procaccia algorithm [12] captures this intuition by calculating the correlation dimension. First, calculate the correlation integral:

$$C(r) = \frac{2}{N(N-1)} \sum_{i=1}^{N} \sum_{j>i}^{N} H(r - |x_i - x_j|), \tag{16}$$

where $H(x)$ is a step function with $H(x) = 1$ for $x > 0$ and $H(x) = 0$ for $x < 0$. This measures the probability that any two points fall within the ball of radius $r$. Then define

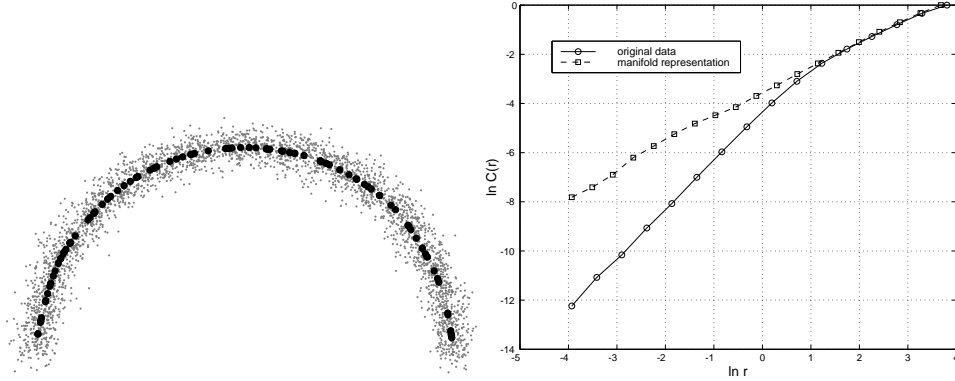

Figure 2: The semicircle. (a) $N = 3150$ points randomly scattered around a semicircle of radius $R = 20$ by a normal process with $\sigma = 1$ and the final positions of 100 manifold points. (b) Log log plot of $C(r)$ vs $r$ for both the manifold points (squares) and the original data set (circles).

the correlation dimension at length scale $r$ as the slope on the log log plot.

$$d_{corr}(r) = \frac{d \log C(r)}{d \log r}. \tag{17}$$

For points lying on a manifold the slope remains constant and the dimensionality is fixed, while the correlation dimension of the original data set quickly approaches that of the embedding space as we decrease the length scale. Note that the slope at large length scales always tends to decrease due to finite span of the data and curvature effects and therefore does not provide a reliable estimator of intrinsic dimensionality.

## 5 Examples

### 5.1 Semi-Circle

We have randomly generated $N = 3150$ data points scattered by a normal distribution with $\sigma = 1$ around a semi-circle of radius $R = 20$ (Figure 2a). Then we ran the algorithm with $K = 100$ and $\lambda = 8$, and terminated the iterative algorithm once the precision $\epsilon = 0.1$ had been reached. The resulting manifold is depicted in red.

To test the quality of our solution, we calculated the correlation dimension as a function of spatial scale for both the manifold points and the original data set (Figure 2b). As one can see, the manifold solution is of fixed dimensionality (the slope remains constant), while the original data set exhibits varying dimensionality. One should also note that the manifold points have $d_{corr}(r) = 1$ well into the territory where the original data set becomes two dimensional. This is what we should expect: at a given information level (in this case, $I = 2.8$ bits), the information about the second (local) degree of freedom is lost, and the resulting structure is one dimensional.

A note about the parameters. Letting $K \to \infty$ does not alter the solution. The information $I$ and distortion $D$ remain the same, and the additional points $\gamma_k$ also fall on the semi-circle and are simple interpolations between the original manifold points. This allows us to claim that what we have found *is* a manifold, and not an agglomeration of clustering centers. Second, varying $\lambda$ changes the information resolution $I(\lambda)$: for small $\lambda$ (high information rate) the local structure becomes important. At high information rate the solution undergoes

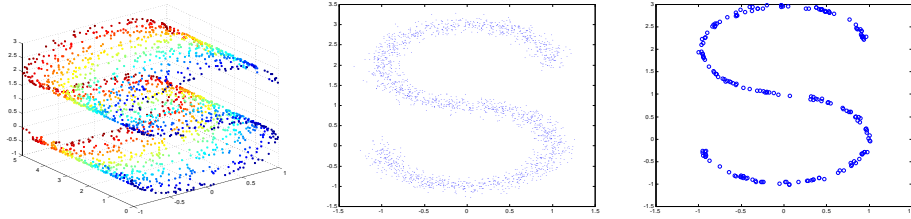

Figure 3: S-shaped sheet in 3D. (a) $N = 2000$ random points on a surface of an S-shaped sheet in 3D. (b) Normal noise added. XY-plane projection of the data. (c) Optimal manifold points in 3D, projected onto an XY plane for easy visualization.

a phase transition, and the resulting manifold becomes two dimensional to take into account the local structure. Alternatively, if we take $\lambda \to \infty$, the cost of information rate becomes very high and the whole manifold collapses to a single point (becomes zero dimensional).

## 5.2   S-surface

Here we took $N = 2000$ points covering an S-shaped sheet in three dimensions (Figure 3a), and then scattered the position of each point by adding Gaussian noise. The resulting manifold is difficult to visualize in three dimensions, so we provided its projection onto an XY plane for an illustrative purpose (Figure 3b). After running our algorithm we have recovered the original structure of the manifold (Figure 3c).

## 6   Discussion

The problem of finding low dimensional manifolds in high dimensional data requires regularization to avoid hgihly folded, Peano curve like solutions which are low dimensional in the mathematical sense but fail to capture our geometric intuition. Rather than constraining geometrical features of the manifold (e.g., the curvature) we have constrained the mutual information between positions on the manifold and positions in the original data space, and this is invariant to all invertible coordinate transformations in either space. This approach enforces "smoothness" of the manifold only implicitly, but nonetheless seems to work. Our information theoretic approach has considerable generality relative to methods based on specific smoothing criteria, but requires a separate algorithm, such as LLE, to give the manifold points curvilinear coordinates. For data points not in the original data set, equations (9-10) and (13-14) provide the mapping onto the manifold. Eqn. (7) gives the probability distribution over the latent variable, known in the density modeling literature as "the prior."

The running time of the algorithm is linear in $N$. This compares favorably with other methods and makes it particularly attractive for very large data sets. The number of manifold points $K$ usually is chosen as large as possible, given the computational constraints, to have a dense sampling of the manifold. However, a value of $K << N$ is often sufficient, since $D(\lambda, K) \to D(\lambda)$ and $I(\lambda, K) \to I(\lambda)$ approach their limits rather quickly (the convergence improves for large $\lambda$ and deteriorates for small $\lambda$). In the example of a semi-circle, the value of $K = 30$ was sufficient at the compression level of $I = 2.8$ bits. In general, the threshold value for $K$ scales exponentially with the latent dimensionality (rather than with the dimensionality of the embedding space).

The choice of $\lambda$ depends on the desired information resolution, since $I$ depends on $\lambda$. Ideally, one should plot the function $I(\lambda)$ and then choose the region of interest. $I(\lambda)$

is a monotonically decreasing function, with the kinks corresponding to phase transitions where the optimal manifold abruptly changes its dimensionality. In practice, we may want to run the algorithm only for a few choices of $\lambda$, and we would like to start with values that are most likely to correspond to a low dimensional latent variable representation. In this case, as a rule of thumb, we choose $\lambda$ smaller, but on the order of the largest linear dimension (i.e. $\sqrt{\lambda/2} \sim L_{max}$). The dependence of the optimal manifold $\mathcal{M}(I)$ on information resolution reflects the multi-scale nature of the data and should not be taken as a shortcoming.

## References

[1] Bregler, C. & Omohundro, S. (1995) Nonlinear image interpolation using manifold learning. *Advances in Neural Information Processing Systems 7*. MIT Press.

[2] Hastie, T. & Stuetzle, W. (1989) Principal curves. *Journal of the American Statistical Association*, 84(406), 502-516.

[3] Roweis, S. & Saul, L. (2000) Nonlinear dimensionality reduction by locally linear embedding. *Science*, *290*, 2323–2326.

[4] Tenenbaum, J., de Silva, V., & Langford, J. (2000) A global geometric framework for nonlinear dimensionality reduction. *Science*, *290* , 2319–2323.

[5] Hotelling, H. (1933) Analysis of a complex of statistical variables into principal components. *Journal of Educational Psychology*, 24:417-441,498-520.

[6] Bishop, C., Svensen, M. & Williams, C. (1998) GTM: The generative topographic mapping. *Neural Computation*,*10*, 215–234.

[7] Brand, M. (2003) Charting a manifold. *Advances in Neural Information Processing Systems 15*. MIT Press.

[8] Scholkopf, B., Smola, A. & Muller K-R. (1998) Nonlinear component analysis as a kernel eigenvalue problem. *Neural Computation*, 10, 1299-1319.

[9] Kramer, M. (1991) Nonlinear principal component analysis using autoassociative neural networks. *AIChE Journal*, 37, 233-243.

[10] Belkin M. & Niyogi P. (2003) Laplacian eigenmaps for dimensionality reduction and data representation. *Neural Computation*, 15(6), 1373-1396.

[11] Blahut, R. (1972) Computation of channel capacity and rate distortion function. *IEEE Trans. Inform. Theory*, IT-18, 460-473.

[12] Grassberger, P., & Procaccia, I. (1983) Characterization of strange attractors. *Physical Review Letters*, 50, 346-349.
